# An MEG Study of Response Latency and Variability in the Human Visual System During a Visual-Motor Integration Task

**Akaysha C. Tang**
Dept. of Psychology
University of New Mexico
Albuquerque, NM 87131
*akaysha@unm.edu*

**Barak A. Pearlmutter**
Dept. of Computer Science
University of New Mexico
Albuquerque, NM 87131
*bap@cs.unm.edu*

**Tim A. Hely**
Santa Fe Institute
1399 Hyde Park Road
Santa Fe, NM 87501
*timhely@santafe.edu*

**Michael Zibulevsky**
Dept. of Computer Science
University of New Mexico
Albuquerque, NM 87131
*michael@cs.unm.edu*

**Michael P. Weisend**
VA Medical Center
1501 San Pedro SE
Albuquerque, NM 87108
*mweisend@unm.edu*

## Abstract

Human reaction times during sensory-motor tasks vary considerably. To begin to understand how this variability arises, we examined neuronal populational response time variability at early versus late visual processing stages. The conventional view is that precise temporal information is gradually lost as information is passed through a layered network of mean-rate "units." We tested in humans whether neuronal populations at different processing stages behave like mean-rate "units". A blind source separation algorithm was applied to MEG signals from sensory-motor integration tasks. Response time latency and variability for multiple visual sources were estimated by detecting single-trial stimulus-locked events for each source. In two subjects tested on four visual reaction time tasks, we reliably identified sources belonging to early and late visual processing stages. The standard deviation of response latency was smaller for early rather than late processing stages. This supports the hypothesis that human populational response time variability increases from early to late visual processing stages.

## 1 Introduction

In many situations, precise timing of a motor output is essential for successful task completion. Somehow the reliability in the output timing is related to the reliability of the underlying neural systems associated with different stages of processing. Recent literature from animal studies suggests that individual neurons from different brain regions and different species can be surprising reliable [1, 2, 5, 7–9, 14, 17, 18],

on the order of a few milliseconds. Due to the low spatial resolution of electroencephalography (EEG) and the requirement of signal averaging due to noisiness of magnetoencephalography (MEG), *in vivo* measurement of human *populational* response time variability from different processing stages has not been available.

In four visual reaction time (RT) tasks, we estimated neuronal response time variability at different visual processing stages using MEG. One major obstacle that has prevented the analysis of response timing variability using MEG before is the relative weakness of the brain's magnetic signals (100fT) compared to noise in a shielded environment (magnetized lung contaminants: $10^6$fT; abdominal currents $10^5$fT; cardiogram and oculogram: $10^4$fT; epileptic and spontaneous activity: $10^3$fT) and in the sensors (10fT) [13]. Consequently, neuronal responses evoked during cognitive tasks often require signal averaging across many trials, making analysis of single-trial response times unfeasible.

Recently, Bell-Sejnowski Infomax [1995] and Fast ICA [10] algorithms have been used successfully to isolate and remove major artifacts from EEG and MEG data [11, 15, 20]. These methods greatly increase the effective signal-to-noise ratio and make single-trial analysis of EEG data feasible [12]. Here, we applied a Second-Order Blind Identification algorithm (SOBI) [4] (another blind source separation, or *BSS*, algorithm) to MEG data to find out whether *populational* response variability changes from early to late visual processing stages.

## 2  Methods

### 2.1  Experimental Design

Two volunteer normal subjects (females, right handed) with normal or corrected-to-normal visual acuity and binocular vision participated in four different visual RT tasks. Subjects gave informed consent prior to the experimental procedure. During each task we recorded continuous MEG signals at a 300Hz sampling rate with a band-pass filter of 1–100Hz using a 122 channel Neuromag-122.

In all four tasks, the subject was presented with a pair of abstract color patterns, one in the left and the other in the right visual field. One of the two patterns was a target pattern. The subject pressed either a left or right mouse button to indicate on which side the target pattern was presented. When a correct response was given, a low or high frequency tone was presented binaurally following respectively a correct or wrong response. The definition of the target pattern varied in the four tasks and was used to control task difficulty which ranged from easy (task 1) to more difficult (task 4) with increasing RTs. (The specific differences among the four tasks are not important for the analysis which follows and are not discussed further.)

In this study we focus on the one element that all tasks have in common, i.e. activation of multiple visual areas along the visual pathways. Our goal is to identify visual neuronal sources activated in all four visual RT tasks and to measure and compare response time variability between neuronal sources associated with early and later visual processing stages. Specifically, we test the hypothesis that populational neuronal response times increase from early to later visual processing stages.

### 2.2  Source Separation Using SOBI

In MEG, magnetic activity from different neuronal populations is observed by many sensors arranged around the subject's head. Each sensor responds to a mixture of the signals emitted by multiple sources. We used the Second-Order Blind Identi-

fication algorithm (SOBI) [4] (a BSS algorithm) to simultaneously separate neuromagnetic responses from different neuronal populations associated with different stages of visual processing. Responses from different neuronal populations will be referred to as *source responses* and the neuronal populations that give rise to these responses will be referred to as *neuronal sources* or simply *sources*. These neuronal sources often, but not always, consist of a spatially contiguous population of neurons. BSS separates the measured sensor signals into maximally independent components, each having its own spatial map. Previously we have shown that some of these BSS separated components correspond to noise sources, and many others correspond to neuronal sources [19].

To establish the identity of the components, we analyzed both temporal and spatial properties of the BSS separated components. Their temporal properties are displayed using *MEG images*, similar to the ERP images described by [12] but without smoothing across trials. These MEG images show stimulus or response locked responses across many trials in a map, from which response latencies across all displayed trials can be observed with a glance. The spatial properties of the separated components are displayed using a *field map* that shows the sensor projection of a given component. The intensity at each point on the field map indicates how strongly this component influences the sensor at this location.

The correspondence between the separated components and neuronal populational responses at different visual processing stages were established by considering both spatial and temporal properties of the separated components [19]. For example, a component was identified as an early visual neuronal source if and only if (1) the field pattern, or the sensor projection, of the separated component showed a focal response over the occipital lobe, and (2) the ERP image showed visual stimulus locked responses with latencies shorter than all other visual components and falling within the range of early visual responses reported in studies using other methods. *Only* those components consistent both spatially and temporally with known neurophysiology and neuroanatomy were identified as neuronal sources.

## 2.3 Single Event Detection and Response Latency Estimation

For all established visual components we calculated the single-trial response latency as follows. First, a detection window was defined using the stimulus-triggered average (STA). The beginning of the detection window was defined by the time at which the STA first exceeded the range of baseline fluctuation. Baseline fluctuation was estimated from the time of stimulus onset for approximately 50ms (the visual response occurred no earlier than 60ms after stimulus onset.) The detection window ended when the STA first returned to the same level as when the detection window began. The detection threshold was determined using a control window with the same width as the detection window, but immediately preceding the detection window. The threshold was adjusted until no more than five false detections occurred within the control window for each ninety trials. We estimated RTs using the leading edge of the response, rather than the time of the peak as this is more robust against noise.

## 3 Results

In both subjects across all four visual RT tasks, SOBI generated components that corresponded to neuronal populational responses associated with early and late stages of visual processing. In both subjects, we identified a single component with a sensor projection at the occipital lobe whose latency was the shortest among all

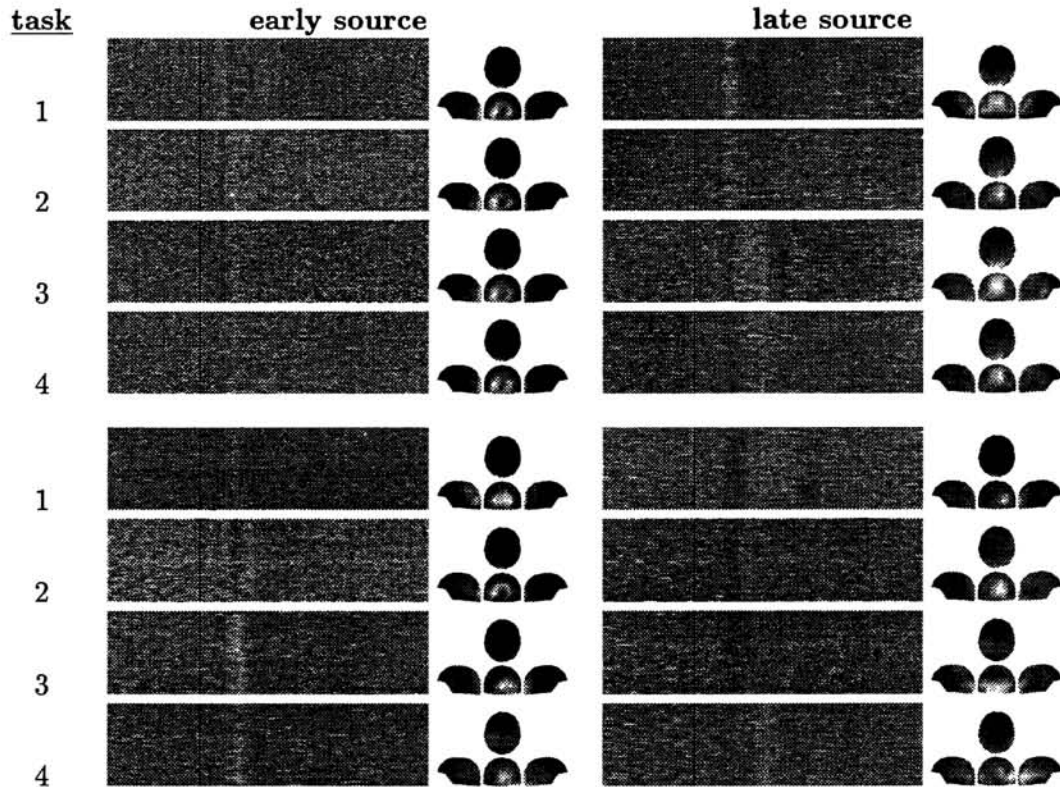

Figure 1: MEG images and field maps for an early and a late source from each
task, for subject 1 (top) and subject 2 (bottom). MEG image pixels are brightness-
coded source strength. Each row of a bitmap is one trial, running 1170ms from left
to right. Vertical bars mark stimulus onset, and 333ms of pre-stimulus activity is
shown. Each panel contains 90 trials. Field map brightness indicates the strength
with which a source activates each of the 61 sensor pairs.

visual stimulus locked components within task and subject (Fig. 1 left). We iden-
tified multiple components that had sensor projections either at occipital-parietal,
occipital-temporal, or temporal lobes, and whose response latencies are longer than
early-stage components within task and subject (Fig. 1 right).

Fig. 2a shows examples of detected single-trial responses for one early and one late
visual component (left: early; right: late) from one task. To minimize false positives,
the detection threshold was set high (allowing 5 false detections out of 90 trials)
at the expense of a low detection rate (15%–67%.) When Gaussian filters were
applied to the raw separated data, the detection rates were increased to 22–91%
(similar results hold but not shown). Fig. 2b shows such detected response time
histograms superimposed on the stimulus triggered average using raw separated
data. One early (top row) and two late visual components (middle and bottom
rows) are plotted for each of the four experiments in subject one. The histogram
width is smallest for early visual components (short mean response latency) and
larger for late visual components (longer latency.)

We computed the standard deviation of component response times as a measure of
response variability. Fig. 2c shows the response variability as a function of mean
response latency for subject one. Early components (solid boxes, shorter mean
latency) have smaller variability (height of the boxes) while late components (dashed
boxes, longer mean latency) have larger variability (height of the boxes). Multiple

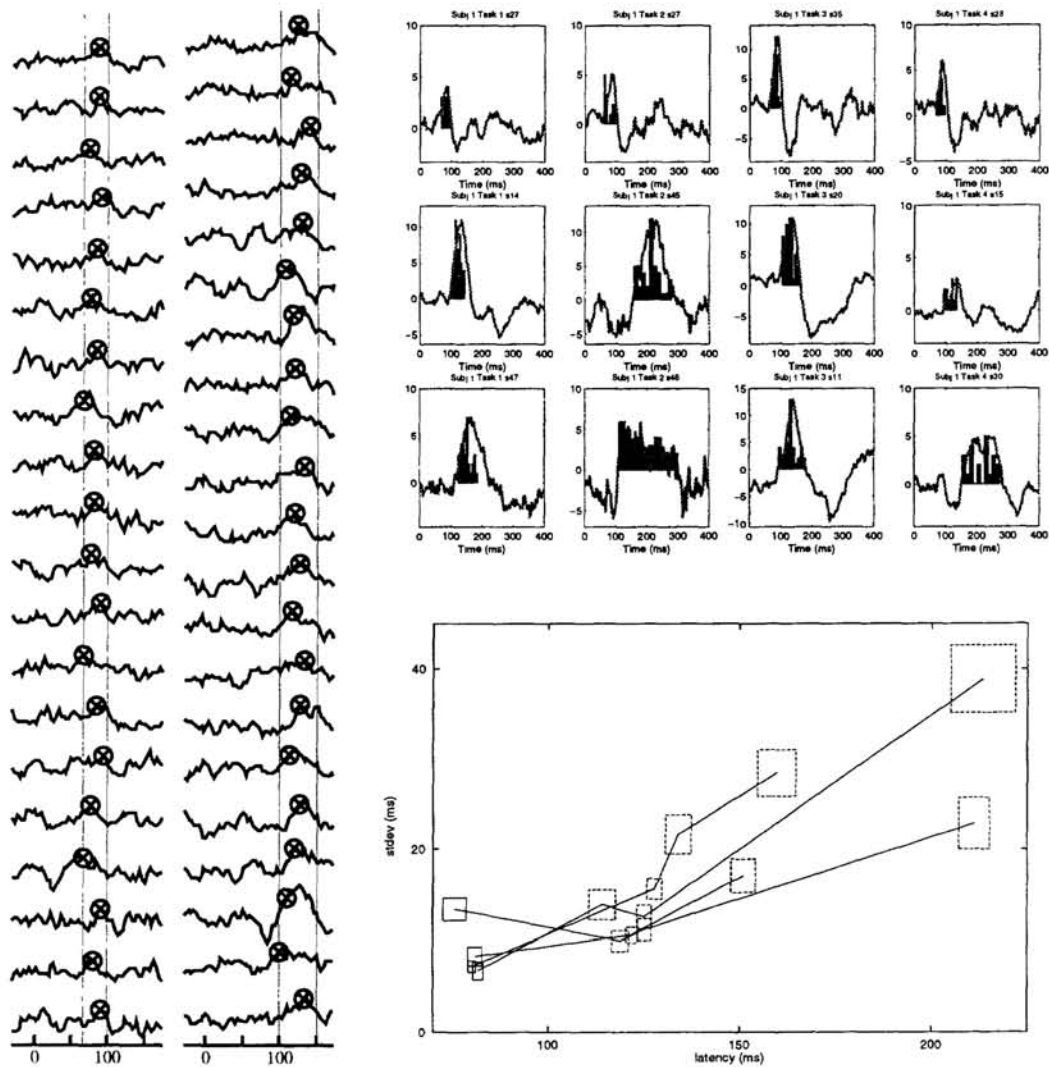

Figure 2: (**a**, left) Response onset was estimated for each trial via threshold crossing within a window of eligibility. (**b**, top right) The stimulus-locked averages for a number of sources overlaid on histograms of response onset times. (**c**, bottom right) Scatter plot of visual components from all experiments on subject 1 showing the standard deviation of the latency (y axis) versus the mean latency (x axis), with the error bars in each direction indicating one standard error in the respective measurement. Lines connect sources from each task.

visual components from each task are connected by a line. Four tasks were shown here. There is a general trend of increasing standard deviation of response times as a function of early-to-late processing stages (increasing mean latency from left to right). For the early visual components the standard deviation ranges from $6.6\pm0.63$ms to $13.4\pm1.23$ms, and for the late visual components, from $9.9\pm0.86$ms to $38.8\pm3.73$ms ($t = 3.565$, $p = 0.005$.)

## 4 Discussion

By applying SOBI to MEG data from four visual RT tasks, we separated components corresponding to neuronal populational responses associated with early and

later stage visual processing in both subjects across all tasks. We performed single-trial RT detection on these early- and late-stage components and estimated both the mean and stdev of their response latency. We found that variability of the populational response latency increased from early to late processing stages.

These results contrast with single neuron recordings obtained previously. In early and late visual processing stages, the rise time of mean firing rate in single units remained constant, suggesting an invariance in RT variability [16]. Characterizing the precise relationship between single neuron and populational response reliability is difficult without careful simulations or simultaneous single unit and MEG recording. However, some major differences exist between the two types of studies. While MEG is more likely to sample a larger neuronal population, single unit studies are more likely to be selective to those neurons that are already highly reliable in their responses to stimulus presentation. It is possible that the most reliable neurons at both the early and late processing stages are equally reliable while large differences exist between the early and late stages for the low reliability neurons.

Previously, ICA algorithms have been used successfully to separate out various noise and neuronal sources in MEG data [19, 20]. Here we show that SOBI can also be used to separate different neuronal sources, particularly those associated with different processing stages. The SOBI algorithm assumes that the components are independent across multiple time scales and attempts to minimize the temporal correlation at these time scales. Although neuronal sources at different stages of processing are not completely independent as assumed in SOBI's derivation, BSS algorithms of this sort are quite robust even when the underlying assumptions are not fully met [6], *i.e.* the goodness of the separation is not significantly affected. The ultimate reality check should come from satisfying physiological and anatomical constraints derived from prior knowledge of the neural system under study. This was carried out for our analysis. Firstly, the average response latencies of the separated components fell within the range of latencies reported in MEG studies using conventional source modeling methods. Secondly, the spatial patterns of sensor responses to these separated components are consistent with the known functional anatomy of the visual system.

We have attempted to rule out many confounding factors. Our observed results cannot be accounted for by a higher signal to noise ratio in the early visual responses. The increase in measured onset response time variability from early to late visual processing stages was actually accompanied by an slightly *lower* signal-to-noise ratio among the early components. The number of events detected for the later components were also slightly greater than the earlier components. The higher signal-to-noise ratio at later components should *reduce* noise-induced variability in the later components, which would bias against the hypothesis that later visual responses have greater response time variability. We also found that response duration and detection window size cannot account for the observed differential variabilities. Later visual responses also had gentler onset slopes (as measured by the stimulus-triggered average). Sensor noise unavoidably introduces noise into the response onset detection process. We cannot rule out the possibility that the interaction of the noise with the response onset profiles might give rise to the observed differential variabilities. Similarly, we cannot rule out the possibility that even greater control of the experimental situation, such as better fixation and more effective head restraints, would differentially reduce the observed variabilities. In general, all measured variabilities can only be upper bounds, subject to downward revision as improved instrumentation and experiments become available. It is with this caution in mind that we conclude that response time variability of neuronal populations increases from early to late processing stages in the human visual system.

## Acknowledgments

This research was supported by NSF CAREER award 97-02-311, and by the National Foundation for Functional Brain Imaging.

## References

[1] M. Abeles, H. Bergman, E. Margalit, and E Vaadia. Spatiotemporal firing patterns in the frontal cortex of behaving monkeys. *J. Neurophys.*, 70:1629–1638, 1993.

[2] W. Bair and C. Koch. Temporal precision of spike trains in extrastriate cortex of the behaving macaque monkey. *Neural Computation*, 8(6):1184–1202, 1996.

[3] A. J. Bell and T. J. Sejnowski. An information-maximization approach to blind separation and blind deconvolution. *Neural Computation*, 7(6):1129–1159, 1995.

[4] A. Belouchrani, K. A. Meraim, J.-F. Cardoso, and E. Moulines. Second-order blind separation of correlated sources. In *Proc. Int. Conf. on Digital Sig. Proc.*, pages 346–351, Cyprus, 1993.

[5] M. J. Berry, W. K. Warland, and M. Meister. The structure and precision of retinal spike trains. *Proc. Natl. Acad. Sci. USA*, 94:5411–5416, 1997.

[6] J.-F. Cardoso. Blind signal separation: statistical principles. *Proceedings of the IEEE*, 9(10):2009–2025, October 1998.

[7] R. R. de Ruyter van Steveninck, G. D. Lewen, S. P. Strong, R. Koberle, and W. Bialek. Reproducibility and variability in neural spike trains. *Science*, 275:1805–1808, 1997.

[8] R. C. deCharms and M. M. Merzenich. Primary cortical representation of sounds by the coordination of action-potential timing. *Nature*, 381:610–3, 1996.

[9] M. Gur, A. Beylin, and D. M. Snodderly. Response variability of neurons in primary visual cortex (V1) of alert monkeys. *J. Neurosci.*, 17(8):2914–2920, 1997.

[10] A. Hyvarinen and E. Oja. A fast fixed-point algorithm for independent component analysis. *Neural Computation*, 9(7), October 1997.

[11] T.-P. Jung, C. Humphries, T.-W. Lee, M. J. McKeown, V. Iragui, S. Makeig, and T. J. Sejnowski. Removing electroencephalographic artifacts by blind source separation. *Psychophysiology*, 1999. In Press.

[12] T.-P. Jung, S. Makeig, M. Westerfield, J. Townsend, E. Courchesne, and T. J. Sejnowski. Analyzing and visualizing single-trial event-related potentials. In *Advances in Neural Information Processing Systems 11*, pages 118–124. MIT Press, 1999.

[13] J. D. Lewine and W. W. Orrison, II. Magnetoencephalography and magnetic source imaging. In *Functional Brain Imaging*, pages 369–417. Mosby, St. Louis, 1995.

[14] Z. F. Mainen and T. J. Sejnowski. Reliability of spike timing in neocortical neurons. *Science*, 268:1503–1506, 1995.

[15] S. Makeig, T.-P. Jung, A. J. Bell, D. Ghahremani, and T. J. Sejnowski. Blind separation of auditory event-related brain responses into independent components. *Proc. Nat. Acad. Sci.*, 94:10979–84, 1997.

[16] P. Marsalek, C. Koch, and J. Maunsell. On the relationship between synaptic input and spike output jitter in individual neurons. *Proc. Natl. Acad. Sci.*, 94:735–40, 1997.

[17] D. S. Reich, J. D. Victor, B. W. Knight, and T. Ozaki. Response variability and timing precision of neuronal spike trains *in vivo*. *J. Neurophys.*, 77:2836–2841, 1997.

[18] A. C. Tang, A. M. Bartels, and T. J. Sejnowksi. Effects of cholinergic modulation on responses of neocortical neurons to fluctuating inputs. *Cereb. Cortex*, 7:502–9, 1997.

[19] A. C. Tang, B. A. Pearlmutter, M. Zibulevsky, and R. Loring. Response time variability in the human sensory and motor systems. In *Computational Neuroscience*, 1999. To appear as a special issue of *Neurocomputing*.

[20] R. Vigário, V. Jousmäki, M. Hämäläinen, R. Hari, and E. Oja. Independent component analysis for identification of artifacts in magnetoencephalographic recordings. In *Advances in Neural Information Processing Systems 10*. MIT Press, 1998.
